# Weighted Sums of Random Kitchen Sinks: Replacing minimization with randomization in learning

**Paper #858**

## Abstract

Randomized neural networks are immortalized in this AI Koan:

> *In the days when Sussman was a novice, Minsky once came to him as he sat hacking at the PDP-6.*
>
> *"What are you doing?" asked Minsky. "I am training a randomly wired neural net to play tic-tac-toe," Sussman replied. "Why is the net wired randomly?" asked Minsky. Sussman replied, "I do not want it to have any preconceptions of how to play."*
>
> *Minsky then shut his eyes. "Why do you close your eyes?" Sussman asked his teacher. "So that the room will be empty," replied Minsky. At that moment, Sussman was enlightened.*

We analyze shallow random networks with the help of concentration of measure inequalities. Specifically, we consider architectures that compute a weighted sum of their inputs after passing them through a bank of arbitrary randomized nonlinearities. We identify conditions under which these networks exhibit good classification performance, and bound their test error in terms of the size of the dataset and the number of random nonlinearities.

## 1  Introduction

In the earliest days of artificial intelligence, the bottom-most layer of neural networks consisted of randomly connected "associator units" that computed random binary functions of their inputs [1]. These randomized shallow networks have largely been superceded by optimally, or nearly optimally, tuned shallow architectures such as weighted sums of positive definite kernels (as in Support Vector Machines), or weigted sums of weak classifiers (as in Adaboost). But recently, architectures that randomly transform their inputs have been resurfacing in the machine learning community [2, 3, 4, 5], largely motivated by the fact that randomization is computationally cheaper than optimization. With the help of concentration of measure inequalities on function spaces, we show that training a shallow architecture by randomly choosing the nonlinearities in the first layer results in a classifier that is not much worse than one constructed by optimally tuning the nonlinearities. The main technical contributions of the paper are an approximation error bound (Lemma 1), and a synthesis of known techniques from learning theory to analyze random shallow networks.

Consider the problem of fitting a function $f : \mathcal{X} \to \mathbb{R}$ to a training data set of $m$ input-output pairs $\{x_i, y_i\}_{i=1...m}$, drawn iid from some unknown distribution $P(x, y)$, with $x_i \in \mathcal{X}$ and $y_i \in \pm 1$. The fitting problem consists of finding an $f$ that minimizes the empirical risk

$$\mathbf{R}_{emp}[f] \equiv \frac{1}{m} \sum_{i=1}^{m} c(f(x_i), y_i). \tag{1}$$

The loss $c(y, y')$ penalizes the deviation between the prediction $f(x)$ and the label $y$. Popular choices for $c$ are the hinge loss, $\max(0, 1 - yy')$, used in the Support Vector Machine [6], the exponential loss, $e^{-yy'}$, used in Adaboost [7, 8], and the quadratic loss, $(y - y')^2$, used in matching pursuit [9] and regularized least squares classification [10].

Similarly to kernel machines and Adaboost, we will consider functions of the form $f(x) = \sum_{i=1}^{\infty} \alpha(w_i)\phi(x; w_i)$ or $f(x) = \int \alpha(w)\phi(x; w) \; dw$, where feature functions $\phi : \mathcal{X} \times \Omega \to \mathbb{R}$, parameterized by some vector $w \in \Omega$, are weighted by a function $\alpha : \Omega \to \mathbb{R}$. In kernel machines, the feature functions $\phi$ are the eigenfunctions of a positive definite kernel $k$, and in Adaboost they are typically decision trees or stumps. Adaboost [8, 7] and matching pursuit [11, 9] find approximate empirical risk minimizer over this class of functions by greedily minimizing over a finite number of scalar weights $\alpha$ and parameter vectors $w$ jointly:

$$\underset{\substack{w_1, \ldots, w_K \in \Omega \\ \alpha \in \mathcal{A}}}{\text{minimize}} \quad \mathbf{R}_{emp}\left[\sum_{k=1}^{K} \phi(x; w_k)\alpha_k\right]. \tag{2}$$

But it is also possible to *randomize* over $w$ and minimize over $\alpha$. Rather than jointly optimizing over $\alpha$ and $w$, the following algorithm first draws the parameters of the nonlinearities randomly from a pre-specificied distribution $p$. Then with $w$ fixed, it fits the weights $\alpha$ optimally via a simple convex optimization:

---

**Algorithm 1** The Weighted Sum of Random Kitchen Sinks fitting procedure.

---

**Input:** A dataset $\{x_i, y_i\}_{i=1\ldots m}$ of $m$ points, a bounded feature function $|\phi(x; w)| \leq 1$, an integer $K$, a scalar $C$, and a probability distribution $p(w)$ on the parameters of $\phi$.

**Output:** A function $\hat{f}(x) = \sum_{k=1}^{K} \phi(x; w_k)\alpha_k$.

Draw $w_1, \ldots, w_K$ iid from $p$.

Featurize the input: $z_i \leftarrow [\phi(x_i; w_1), \ldots, \phi(x_i; w_K)]^\top$.

With $w$ fixed, solve the empirical risk minimization problem

$$\underset{\alpha \in \mathbb{R}^K}{\text{minimize}} \quad \frac{1}{m} \sum_{i=1}^{m} c\left(\alpha^\top z_i, \; y_i\right) \tag{3}$$

$$\text{s.t.} \quad \|\alpha\|_\infty \leq C/K. \tag{4}$$

---

In pratice, we let $C$ be large enough that the constraint (4) remains inactive. The when $c$ is the quadratic loss, the minimization (3) is simple linear least squares, and when $c$ is the hinge loss, it amounts of fitting a *linear* SVM to a dataset of $m$ $K$-dimensional feature vectors.

Randomly setting the nonlinearities is appealing for several reasons. First, the fitting procedure is simple: Algorithm 1 can be implemented in a few lines of MATLAB code even for complex feature functions $\phi$, whereas fitting nonlinearities with Adaboost requires much more care. This flexibility allows practioners to experiment with a wide variety of nonlinear feature fuctions without first having to devise fitting procedures for them. Second, the algorithm is fast: experiments show between one and three orders of magnitude speedup over Adaboost. On the down side, one might expect to have to tune the sampling distribution $p$ for each dataset. But in practice, we find that to obtain accuracies that are competitive with Adaboost, the same sampling distribution can be used for all the datasets we considered if the coordinates of the data are first zero-meaned and rescaled to unit variance.

Formally, we show that Algorithm 1 returns a function that has low true risk. The true risk of a function $f$ is

$$\mathbf{R}[f] \equiv \underset{(x,y)\sim P}{\mathbb{E}} c(f(x), y), \tag{5}$$

and measures the expected loss of $f$ on as-yet-unseen test points, assuming these test points are generated from the same distribution that generated the training data. The following theorem states that with very high probability, Algorithm 1 returns a function whose true risk is near the lowest true risk attainable by functions in the class $\mathcal{F}_p$ defined below:

**Theorem 1** (Main result). *Let $p$ be a distribution on $\Omega$, and let $\phi$ satisfy $\sup_{x,w} |\phi(x; w)| \leq 1$. Define the set*

$$\mathcal{F}_p \equiv \left\{ f(x) = \int_\Omega \alpha(w)\phi(x; w) \; dw \; \middle| \; |\alpha(w)| \leq Cp(w) \right\}. \tag{6}$$

Suppose $c(y, y') = c(yy')$, with $c(yy')$ L-Lipschitz. Then for any $\delta > 0$, if the training data $\{x_i, y_i\}_{i=1...m}$ are drawn iid from some distribution $P$, Algorithm 1 returns a function $\hat{f}$ that satisfies

$$\mathbf{R}[\hat{f}] - \min_{f \in \mathcal{F}_p} \mathbf{R}[f] \leq O\left(\left(\frac{1}{\sqrt{m}} + \frac{1}{\sqrt{K}}\right) LC\sqrt{\log \frac{1}{\delta}}\right) \tag{7}$$

with probability at least $1 - 2\delta$ over the training dataset and the choice of the parameters $w_1, \ldots, w_K$.

Note that the dependence on $\delta$ in the bound is logarithmic, so even small $\delta$'s do not cause the bound to blow up. The set $\mathcal{F}_p$ is a rich class of functions. It consists of functions whose weights $\alpha(w)$ decays more rapidly than the given sampling distribution $p$. For example, when $\phi(x; w)$ are sinusoids with frequency $w$, $\mathcal{F}_p$ is the set of all functions whose Fourier transforms decay faster than $C\,p(w)$.

We prove the theorem in the next section, and demonstrate the algorithm on some sample datasets in Section 4. The proof of the theorem provides explicit values for the constants in the big O notation.

## 2 Proof of the Main Theorem

Algorithm 1 returns a function that lies in the random set

$$\hat{\mathcal{F}}_w \equiv \left\{ f(x) = \sum_{k=1}^{K} \alpha_k \phi(x; w_k) \,\middle|\, |\alpha_k| \leq \frac{C}{K} \right\}. \tag{8}$$

The bound in the main theorem can be decomposed in a standard way into two bounds:

1. An approximation error bound that shows that the lowest true risk attainable by a function in $\hat{\mathcal{F}}_w$ is not much larger than the lowest true risk attainable in $\mathcal{F}_p$ (Lemma 2).

2. An estimation error bound that shows that the true risk of every function in $\hat{\mathcal{F}}_w$ is close to its empirical risk (Lemma 3).

The following Lemma is helpful in bounding the approximation error:

**Lemma 1.** *Let $\mu$ be a measure on $\mathcal{X}$, and $f^*$ a function in $\mathcal{F}_p$. If $w_1, \ldots, w_K$ are drawn iid from $p$, then for any $\delta > 0$, with probability at least $1 - \delta$ over $w_1, \ldots, w_K$, there exists a function $\hat{f} \in \hat{\mathcal{F}}_w$ so that*

$$\sqrt{\int_{\mathcal{X}} \left(\hat{f}(x) - f^*(x)\right)^2 d\mu(x)} \leq \frac{C}{\sqrt{K}}\left(1 + \sqrt{2\log\frac{1}{\delta}}\right). \tag{9}$$

The proof relies on Lemma 4 of the Appendix, which states that the average of bounded vectors in a Hilbert space concentrates towards its expectation in the Hilbert norm exponentially fast.

*Proof.* Since $f^* \in \mathcal{F}_p$, we can write $f^*(x) = \int_\Omega \alpha(w)\phi(x; w)\, dw$. Construct the functions $f_k = \beta_k \phi(\cdot; w_k)$, $k = 1 \ldots K$, with $\beta_k \equiv \frac{\alpha(\omega_k)}{p(\omega_k)}$, so that $\mathbb{E}\, f_k = f^*$. Let $\hat{f}(x) = \sum_{k=1}^{K} \frac{\beta_k}{K}\phi(x; \omega_k)$ be the sample average of these functions. Then $\hat{f} \in \hat{\mathcal{F}}_w$ because $|\beta_k/K| \leq C/K$. Also, under the inner product $\langle f, g \rangle = \int f(x)g(x)\, d\mu(x)$, $\|\beta_k \phi(\cdot; w_k)\| \leq C$. The Lemma follows by applying Lemma 4 to $f_1, \ldots, f_K$ under this inner product. $\square$

**Lemma 2** (Bound on the approximation error). *Suppose $c(y, y')$ is L-Lipschitz in its first argument. Let $f^*$ be a fixed function in $\mathcal{F}_p$. If $w_1, \ldots, w_K$ are drawn iid from $p$, then for any $\delta > 0$, with probability at least $1 - \delta$ over $w_1, \ldots, w_K$, there exists a function $\hat{f} \in \hat{\mathcal{F}}_w$ that satisfies*

$$\mathbf{R}[\hat{f}] \leq \mathbf{R}[f^*] + \frac{LC}{\sqrt{K}}\left(1 + \sqrt{2\log\frac{1}{\delta}}\right). \tag{10}$$

*Proof.* For any two functions $f$ and $g$, the Lipschitz condition on $c$ followed by the concavity of square root gives

$$\mathbf{R}[f] - \mathbf{R}[g] = \mathbb{E}\, c(f(x), y) - c(g(x), y) \leq \mathbb{E}\, |c(f(x), y) - c(g(x), y)| \tag{11}$$

$$\leq L\; \mathbb{E}\, |f(x) - g(x)| \leq L\sqrt{\mathbb{E}(f(x) - g(x))^2}. \tag{12}$$

The lemma then follows from Lemma 1. $\qquad\square$

Next, we rely on a standard result from statistical learning theory to show that for a given choice of $w_1, \ldots, w_K$ the empirical risk of every function in $\hat{\mathcal{F}}_w$ is close to its true risk.

**Lemma 3** (Bound on the estimation error). *Suppose $c(y, y') = c(yy')$, with $c(yy')$ $L$-Lipschitz. Let $w_1, \cdots, w_K$ be fixed. If $\{x_i, y_i\}_{i=1\ldots m}$ are drawn iid from a fixed distribution, for any $\delta > 0$, with probability at least $1 - \delta$ over the dataset, we have*

$$\forall_{f \in \hat{\mathcal{F}}_w} \quad |\mathbf{R}[f] - \mathbf{R}_{emp}[f]| \leq \frac{1}{\sqrt{m}} \left( 4LC + 2|c(0)| + LC\sqrt{\tfrac{1}{2} \log \tfrac{1}{\delta}} \right). \tag{13}$$

*Proof sketch.* By Hölder, the functions in $\hat{\mathcal{F}}_w$ are bounded above by $C$. The Rademacher complexity of $\hat{\mathcal{F}}_w$ can be shown to be bounded above by $C/\sqrt{m}$ (see the Appendix). The theorem follows by results from [12] which are summarized in Theorem 2 of the Appendix. $\qquad\square$

*Proof of Theorem 1.* Let $f^*$ be a minimizer of $\mathbf{R}$ over $\mathcal{F}_p$, $\hat{f}$ a minimizer of $\mathbf{R}_{emp}$ over $\hat{\mathcal{F}}_w$ (the output of the algorithm), and $\hat{f}^*$ a minimizer of $\mathbf{R}$ over $\hat{\mathcal{F}}_w$. Then

$$\mathbf{R}[\hat{f}] - \mathbf{R}[f^*] = \mathbf{R}[\hat{f}] - \mathbf{R}[\hat{f}^*] + \mathbf{R}[\hat{f}^*] - \mathbf{R}[f^*] \tag{14}$$

$$\leq |\mathbf{R}[\hat{f}] - \mathbf{R}[\hat{f}^*]| + \mathbf{R}[\hat{f}^*] - \mathbf{R}[f^*]. \tag{15}$$

The first term in the right side is an estimation error: By Lemma 3, with probability at least $1 - \delta$, $|\mathbf{R}[\hat{f}^*] - \mathbf{R}_{emp}[\hat{f}^*]| \leq \epsilon_{est}$ and simultaneously, $|\mathbf{R}[\hat{f}] - \mathbf{R}_{emp}[\hat{f}]| \leq \epsilon_{est}$, where $\epsilon_{est}$ is the right side of the bound in Lemma 3. By the optimality of $\hat{f}$, $\mathbf{R}_{emp}[\hat{f}] \leq \mathbf{R}_{emp}[\hat{f}^*]$. Combining these facts gives that with probability at least $1 - \delta$, $|\mathbf{R}[\hat{f}] - \mathbf{R}[\hat{f}^*]| \leq 2\epsilon_{est} = \frac{2}{\sqrt{m}} \left( 4LC + 2|c(0)| + LC\sqrt{\tfrac{1}{2} \log \tfrac{1}{\delta}} \right)$.

The second term in Equation (15) is the approximation error, and by Theorem 1, with probability at least $1 - \delta$, it is bounded above by $\epsilon_{app} = \frac{LC}{\sqrt{K}} \left( 1 + \sqrt{2 \log \tfrac{1}{\delta}} \right)$.

By the union bound, with probability at least $1 - 2\delta$, the right side of Equation (15) is bounded above by $2\epsilon_{est} + \epsilon_{app}$. $\qquad\square$

## 3 Related Work

Greedy algorithms for fitting networks of the form (2) have been analyzed, for example, in [7, 11, 9]. Zhang analyzed greedy algorithms and a randomized algorithm similar to Algorithm 1 for fitting sparse Gaussian processes to data, a more narrow setting than we consider here. He obtained bounds on the expected error for this sparse approximation problem by viewing these methods as stochastic gradient descent.

Approximation error bounds such as that of Maurey [11][Lemma 1], Girosi [13] and Gnecco and Sanguineti [14] rely on random sampling to guarantee the *existence* of good parameters $w_1, \ldots, w_k$, but they require access to the representation of $f^*$ to actually produce these parameters. These approximation bounds cannot be used to guarantee the performance of Algorithm 1 because Algorithm 1 is oblivious of the data when it generates the parameters. Lemma 2 differs from these bounds in that it relies on $f^*$ only to generate the weights $\alpha_1, \ldots, \alpha_K$, but it remains oblivious to $f^*$ when generating the parameters by sampling them from $p$ instead. Furthermore, because $\hat{\mathcal{F}}_w$ is smaller than the classes considered by [11, 14], the approximation error rate in Lemma 1 matches those of existing approximation error bounds.

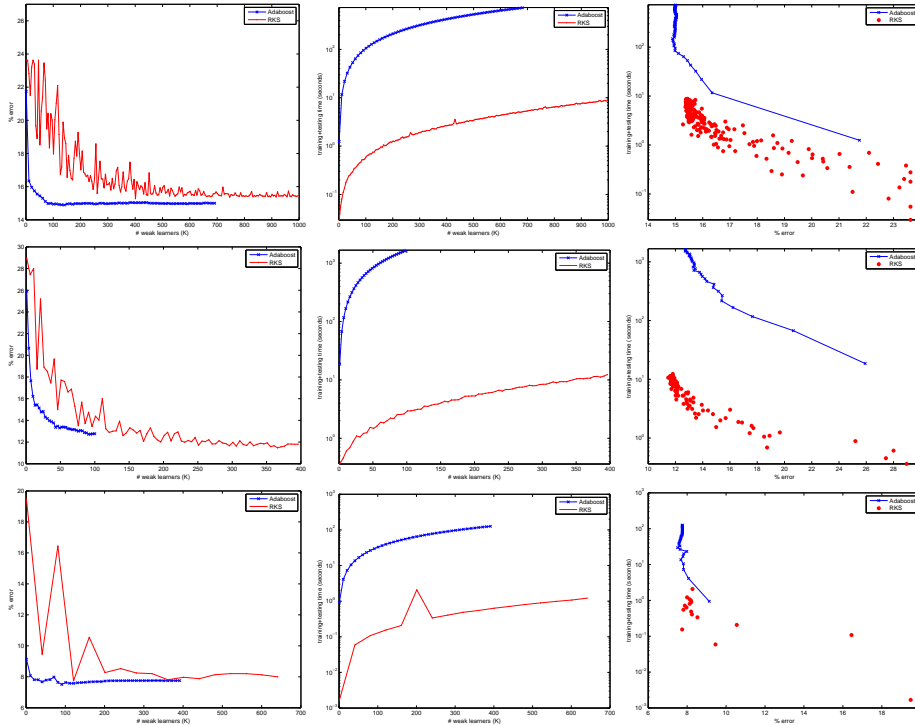

Figure 1: Comparisons between Random Kitchen Sinks and Adaboosted decision stumps on `adult` (first row), `activity` (second row), and `KDDCUP99` (third row). The first column plots test error of each classifier as a function of $K$. The accuracy of Random Kitchen Sinks catches up to that of Adaboost as $K$ grows. The second column plots the total training and testing time as a function of $K$. For a given $K$, Random Kitchen Sinks is between two and three orders of magnitude faster than Adaboost. The third column combines the previous two columns. It plots testing+training time required to achieve a desired error rate. For a given error rate, Random Kitchen Sinks is between one and three orders of magnitude faster than Adaboost.

## 4 Experiments

Since others have already empirically demonstrated the benefits of random featurization [2, 3, 4, 5], we only a present a few illustrations in this section.

We compared Random Kitchen Sinks with Adaboost on three classification problems: The `adult` dataset has roughly 32,000 training instances. Each categorical variable was replaced by a binary indicator variable over the categories, resulting in 123 dimensions per instance. The test set consists of 15,000 instances. `KDDCUP99` is a network intrusion detection problem with roughly 5,000,000 127-dimensional training instances, subsampled to 50,000 instances. The test set consists of  150,000 instances. `activity` is a human activity recognition dataset with 20,0000 223-dimensional instance, of which about 200 are irrelevant for classification. The test set constists of 50,000 instances. The datasets were preprocessed by zero-meaning and rescaling each dimension to unit variance. The feature functions in these experiments were decision stumps $\phi(x; w) = \text{sign}(x_{w_d} - w_t)$, which simply determine whether the $w_d$th dimension of $x$ is smaller or greater than the threshold $w_t$. The sampling distribution $p$ for Random Kitchen Sinks drew the threshold parameter $w_t$ from a normal distribution and the coordinate $w_d$ from a uniform distribution over the coorindates. For some experiments, we could afford to run Random Kitchen Sinks for larger $K$ than Adaboost, and these runs are included in the plots. We used the quadratic loss, but find no substantial differences in quality under the hinge loss (though there is degradation in speed by a factor of 2-10). We used MATLAB optimized versions of Adaboost and Random Kitchen Sinks, and report wall clock time in seconds.

Figure 1 compares the results on these datasets. Adaboost expends considerable effort in choosing the decision stumps and achieves good test accuracy with a few of them. Random Kitchen Sinks

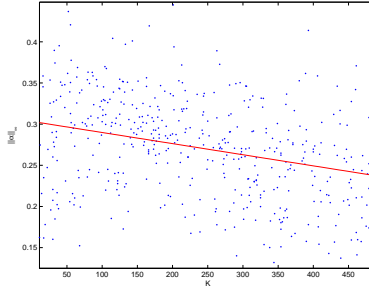

Figure 2: The $L_\infty$ norm of $\alpha$ returned by RKS for 500 different runs of RKS with various settings of $K$ on adult. $\|\alpha\|_\infty$ decays with $K$, which justifies dropping the constraint (4) in practice.

requires more nonlinearities to achieve similar accuracies. But because it is faster than Adaboost, it can produce classifiers that are just as accurate as Adaboost's with more nonlinearities in less total time. In these experiments, Random Kitchen Sinks is almost as accurate as Adaboost but faster by one to three orders of magnitude.

We defer the details of the following experiments to a technical report: As an alternative to Adaboost, we have experimented with conjugate gradient-descent based fitting procedures for (2), and find again that randomly generating the nonlinearities produces equally accurate classifiers using many more nonlinearities but in much less time. We obtain similar results as those of Figure 1 with the random features of [4], and random sigmoidal ridge functions $\phi(x; w) = \sigma(w'x)$,

To simplify the implementation of Random Kitchen Sinks, we ignore the constraint (4) in practice. The scalar $C$ controls the size of $\hat{\mathcal{F}}_w$ and $\mathcal{F}_p$, and to eliminate the constraint, we implicitly set $C$ it to a large value so that the constraint is never tight. However, for the results of this paper to hold, $C$ cannot grow faster than $K$. Figure 2 shows that the $L_\infty$ norm of the unconstrained optimum of (3) for the adult dataset does decays linearly with $K$, so that there exists a $C$ that does not grow with $K$ for which the constraint is never tight, thereby justifying dropping the constraint.

## 5   Discussion and Conclusions

Various hardness of approximation lower bounds for fixed basis functions exist (see, for example [11]). The guarantee in Lemma 1 avoids running afoul of these lower bounds because it does not seek to approximate every function in $\mathcal{F}_p$ simultaneously, but rather only the true risk minimizer with high probability.

It may be surprising that Theorem 1 holds even when the feature functions $\phi$ are nearly orthogonal. The result works because the importance sampling constraint $|\alpha(w)| \le Cp(w)$ ensures that a feature function does not receive a large weight if it is unlikely to be sampled by $p$. When the feature functions are highly linearly dependent, better bounds can be obtained because any $f(x) = \int \alpha(w)\phi(x; w)$ can be rewritten as $f(x) = \int \alpha'(w)\phi(x; w)$ with $|\alpha'|/p \le |\alpha|/p$, improving the importance ratio $C$. This intuition can be formalized via the the Rademacher complexity of $\phi$, a result which we leave for future work.

One may wonder whether Algorithm 1 has good theoretical guarantees on $\mathcal{F}_p$ because $\mathcal{F}_p$ is too small small class of functions. Indeed, when $\phi$ are the Fourier bases, $|\alpha|/p \le C$ implies $\int_\Omega |\alpha(w)| \, dw \le C$, so every function in $\mathcal{F}_p$ has an absolutely integrable Frourier transform. Thus $\mathcal{F}_p$ is smaller than the set considered by Jones [9] for greedy matching pursuit, and for which he obtained an approximation rate of $O(1/\sqrt{K})$. The most reliable way to show that $\mathcal{F}_p$ is rich enough for practical applications is to conduct experiments with real data. The experiment show that $\mathcal{F}_p$ indeed contains good predictors.

The convergence rate for Adaboost [7] is exponentially fast in $K$, which at first appears to be much faster than $1/\sqrt{K}$. However, the base of the exponent is the minimum weighted margin encountered by the algorithm through all iterations, a quantity that is difficult to bound a priori. This makes a direct comparison of the bounds difficult, though we have tried to provide empirical comparisons.

## A  Exponentially Fast Concentration of Averages towards the Mean in a Hilbert Space

**Lemma 4.** *Let* $\mathbf{X} = \{x_1, \cdots, x_K\}$ *be iid random variables in a ball* $\mathcal{H}$ *of radius* $M$ *centered around the origin in a Hilbert space. Denote their average by* $\overline{\mathbf{X}} = \frac{1}{K}\sum_{k=1}^{K} x_k$. *Then for any* $\delta > 0$, *with probability at least* $1 - \delta$,

$$\left\| \overline{\mathbf{X}} - \mathbb{E}\,\overline{\mathbf{X}} \right\| \leq \frac{M}{\sqrt{K}}\left(1 + \sqrt{2\log\tfrac{1}{\delta}}\right). \tag{16}$$

*Proof.* We use McDiarmid's inequality to show that the scalar function $f(\mathbf{X}) = \left\| \overline{\mathbf{X}} - \mathbb{E}_{\mathbf{X}}\,\overline{\mathbf{X}} \right\|$ is concentrated about its mean, which shrinks as $O(1/\sqrt{K})$.

The function $f$ is stable under perturbation of its $i$th argument. Let $\tilde{\mathbf{X}} = \{x_1, \cdots, \tilde{x}_i, \cdots, x_K\}$ be a copy of $\mathbf{X}$ with the $i$th element replaced by an arbitrary element of $\mathcal{H}$. Applying the triangle inequality twice gives

$$|f(\mathbf{X}) - f(\tilde{\mathbf{X}})| = |\|\overline{\mathbf{X}} - \mathbb{E}\,\overline{\mathbf{X}}\| - \|\overline{\tilde{\mathbf{X}}} - \mathbb{E}\,\overline{\mathbf{X}}\|| \leq \|\overline{\mathbf{X}} - \overline{\tilde{\mathbf{X}}}\| \leq \frac{\|x_i - \tilde{x}_i\|}{K} \leq \frac{2M}{K}. \tag{17}$$

To bound the expectation of $f$, use the familiar identity about the variance of the average of iid random variables

$$\mathbb{E}\left\| \overline{\mathbf{X}} - \mathbb{E}\,\overline{\mathbf{X}} \right\|^2 = \frac{1}{K}\left(\mathbb{E}\|x\|^2 - \|\mathbb{E}\,x\|^2\right), \tag{18}$$

in conjunction with Jensen's inequality and the fact that $\|x\| \leq M$ to get

$$\mathbb{E}\,f(\mathbf{X}) \leq \sqrt{\mathbb{E}\,f^2(\mathbf{X})} = \sqrt{\mathbb{E}\left\| \overline{\mathbf{X}} - \mathbb{E}\,\overline{\mathbf{X}} \right\|^2} \leq \frac{M}{\sqrt{K}}. \tag{19}$$

This bound for the expectation of $f$ and McDiarmid's inequality give

$$\Pr_{\mathbf{X}}\left[f(\mathbf{X}) - \frac{M}{\sqrt{K}} \geq \epsilon\right] \leq \Pr_{\mathbf{X}}\left[f(\mathbf{X}) - \mathbb{E}\,f(\mathbf{X}) \geq \epsilon\right] \leq \exp\left(-\frac{K\epsilon^2}{2M^2}\right) \tag{20}$$

To get the final result, set $\delta$ to the right hand side, solve for $\epsilon$, and rearrange. $\qquad\square$

## B  Generalization bounds that use Rademacher complexity

One measure of the size of a class $\mathcal{F}$ of functions is its Rademacher complexity:

$$\mathcal{R}_m[\mathcal{F}] \equiv \mathop{\mathbb{E}}_{\substack{x_1,\cdots,x_m \\ \sigma_1,\cdots,\sigma_m}}\left[\sup_{f\in\mathcal{F}} \frac{1}{m}\sum_{i=1}^{m}\sigma_i f(x_i)\right],$$

The variables $\sigma_1, \cdots, \sigma_m$ are iid Bernouli random variables that take on the value -1 or +1 with equal probability and are independent of $x_1, \ldots, x_m$.

The Rademacher complexity of $\hat{\mathcal{F}}_w$ can be bounded as follows. Define $\mathcal{S} \equiv \left\{\alpha \in \mathbf{R}^K \,\middle|\, \|\alpha\|_\infty \leq \frac{C}{K}\right\}$:

$$\mathcal{R}_m[\hat{\mathcal{F}}_w] = \mathop{\mathbb{E}}_{\sigma,X}\sup_{\alpha\in\mathcal{S}}\left|\frac{1}{m}\sum_{i=1}^{m}\sigma_i\left(\sum_{k=1}^{K}\alpha_k\phi(x_i;\omega_k)\right)\right| = \mathop{\mathbb{E}}_{\sigma,X}\sup_{\alpha\in\mathcal{S}}\left|\sum_{k=1}^{K}\alpha_k\frac{1}{m}\sum_{i=1}^{m}\sigma_i\phi(x_i;\omega_k)\right| \tag{21}$$

$$\leq \mathop{\mathbb{E}}_{\sigma,X}\frac{C}{K}\sum_{k=1}^{K}\left|\frac{1}{m}\sum_{i=1}^{m}\sigma_i\phi(x_i;\omega_k)\right| \leq \mathop{\mathbb{E}}_{X}\frac{C}{K}\sum_{k=1}^{K}\sqrt{\mathop{\mathbb{E}}_{\sigma}\left(\frac{1}{m}\sum_{k=1}^{m}\sigma_i\phi(x_i;\omega_k)\right)^2} \tag{22}$$

$$= \mathop{\mathbb{E}}_{X}\frac{C}{K}\sum_{k=1}^{K}\sqrt{\mathop{\mathbb{E}}_{\sigma}\frac{1}{m^2}\sum_{k=1}^{m}\phi^2(x_i;\omega_k)} \leq \frac{C}{K}\sum_{k=1}^{K}\sqrt{\frac{1}{m}} \leq C/\sqrt{m}, \tag{23}$$

where the first inequality follows by Hölder, the second by the concavity of square root, the third by the fact that conditioned on $\omega$, $\mathbb{E}_\sigma \sigma_i \phi(x_i; \omega) \sigma_j \phi(x_j; \omega) = 0$ when $i \neq j$, and the fourth follows by the boundedness of $\phi$.

The following theorem is a summary of the results from [12]:

**Theorem 2.** *Let $\mathcal{F}$ be a class of bounded functions so that $\sup_x |f(x)| \leq C$ for all $f \in \mathcal{F}$, and suppose $c(y, y') = c(yy')$, with $c(yy')$ L-Lipschitz. Then with probability at least $1 - \delta$ with respect to training samples $\{x_i, y_i\}_m$ drawn from a probabilisty distribution $P$ on $\mathcal{X} \times \{-1, +1\}$, every function in $\mathcal{F}$ satisfies*

$$\mathbf{R}[f] \leq \mathbf{R}_{emp}[f] + 4L\mathcal{R}_m[\mathcal{F}] + \frac{2|c(0)|}{\sqrt{m}} + LC\sqrt{\frac{1}{2m} \log \frac{1}{\delta}}. \tag{24}$$

# References

[1] H. D. Block. The perceptron: a model for brain functioning. *Review of modern physics*, 34:123–135, January 1962.

[2] Y. Amit and D. Geman. Shape quantization and recognition with randomized trees. *Neural Computation*, 9(7):1545–1588, 1997.

[3] F. Moosmann, B. Triggs, and F. Jurie. Randomized clustering forests for building fast and discriminative visual vocabularies. In *Advances in Neural Information Processing Systems (NIPS)*, 2006.

[4] A. Rahimi and B. Recht. Random features for large-scale kernel machines. In *Advances in Neural Information Processing Systems (NIPS)*, 2007.

[5] W. Maass and H. Markram. On the computational power of circuits of spiking neurons. *Journal of Computer and System Sciences*, 69:593–616, December 2004.

[6] E. Osuna, R. Freund, and F. Girosi. Training support vector machines: an application to face detection. In *Computer Vision and Pattern Recognition (CVPR)*, 1997.

[7] R. E. Schapire. The boosting approach to machine learning: An overview. In D. D. Denison, M. H. Hansen, C. Holmes, B. Mallick, and B. Yu, editors, *Nonlinear Estimation and Classification*. Springer, 2003.

[8] J. Friedman, T. Hastie, and R. Tibshirani. Additive logistic regression: a statistical view of boosting. Technical report, Dept. of Statistics, Stanford University, 1998.

[9] L. K. Jones. A simple lemma on greedy approximation in Hilbert space and convergence rates for projection pursuit regression and neural network training. *The Annals of Statistics*, 20(1):608–613, March 1992.

[10] R. Rifkin, G. Yeo, and T. Poggio. Regularized least squares classification. *Advances in Learning Theory: Methods, Model and Applications, NATO Science Series III: Computer and Systems Sciences*, 190, 2003.

[11] A.R. Barron. Universal approximation bounds for superpositions of a sigmoidal function. *IEEE Transactions on Information Theory*, 39:930–945, May 1993.

[12] P. L. Bartlett and S. Mendelson. Rademacher and Gaussian complexities: Risk bounds and structural results. *Journal of Machine Learning Research (JMLR)*, 3:463–482, 2002.

[13] F. Girosi. Approximation error bounds that use VC-bounds. In *International Conference on Neural Networks*, pages 295–302, 1995.

[14] G. Gnecco and M. Sanguineti. Approximation error bounds via Rademacher's complexity. *Applied Mathematical Sciences*, 2(4):153–176, 2008.

